# Learning from the Wisdom of Crowds by Minimax Entropy

**Dengyong Zhou, John C. Platt, Sumit Basu, and Yi Mao**
Microsoft Research
1 Microsoft Way, Redmond, WA 98052
{denzho,jplatt,sumitb,yimao}@microsoft.com

## Abstract

An important way to make large training sets is to gather noisy labels from crowds of nonexperts. We propose a minimax entropy principle to improve the quality of these labels. Our method assumes that labels are generated by a probability distribution over workers, items, and labels. By maximizing the entropy of this distribution, the method naturally infers item confusability and worker expertise. We infer the ground truth by minimizing the entropy of this distribution, which we show minimizes the Kullback-Leibler (KL) divergence between the probability distribution and the unknown truth. We show that a simple coordinate descent scheme can optimize minimax entropy. Empirically, our results are substantially better than previously published methods for the same problem.

## 1 Introduction

There is an increasing interest in using crowdsourcing to collect labels for machine learning [19, 6, 21, 17, 20, 10, 13, 12]. Currently, many companies provide crowdsourcing services. Amazon Mechanical Turk (MTurk) [2] and CrowdFlower [4] are perhaps the most well-known ones. An advantage of crowdsourcing is that we can obtain a large number of labels at the low cost of pennies per label. However, these workers are not experts, so the labels collected from them are often fairly noisy. A fundamental challenge in crowdsourcing is inferring ground truth from noisy labels by a crowd of nonexperts.

When each item is labeled several times by different workers, a straightforward approach is to use the most common label as the true label. From reported experimental results on real crowdsourcing data [19] and our own experience, majority voting performs significantly better on average than individual workers. However, majority voting considers each item independently. When many items are simultaneously labeled, it is reasonable to assume that the performance of a worker is consistent across different items. This assumption underlies the work of Dawid and Skene [5, 18, 19, 11, 17], where each worker is associated with a probabilistic confusion matrix that generates her labels. Each entry of the matrix indicates the probability that items in one class are labeled as another. Given the observed responses, the true labels for each item and the confusion matrices for each worker can be jointly estimated by a maximum likelihood method. The optimization can be implemented by the expectation-maximization (EM) algorithm [7].

Dawid and Skene's method works well in practice. However, their method only contains a per-worker probabilistic confusion model of generating labels. In this paper, we assume a separate probabilistic distribution for each worker-item pair. We propose a novel minimax entropy principle to jointly estimate the distributions and the ground truth given the observed labels by workers in Section 2. The theoretical justification of minimum entropy is given in Section 2.1. To prevent over-fitting, we relax the minimax entropy optimization in Section 3. We describe an easy-to-implement technique to carry out the minimax program in Section 4 and link minimax entropy to a principle of

| | item 1 | item 2 | ... | item $n$ |
|---|---|---|---|---|
| worker 1 | $z_{11}$ | $z_{12}$ | ... | $z_{1n}$ |
| worker 2 | $z_{21}$ | $z_{22}$ | ... | $z_{2n}$ |
| ... | ... | ... | ... | ... |
| worker $m$ | $z_{m1}$ | $z_{m2}$ | ... | $z_{mn}$ |

| | item 1 | item 2 | ... | item $n$ |
|---|---|---|---|---|
| worker 1 | $\pi_{11}$ | $\pi_{12}$ | ... | $\pi_{1n}$ |
| worker 2 | $\pi_{21}$ | $\pi_{22}$ | ... | $\pi_{2n}$ |
| ... | ... | ... | ... | ... |
| worker $m$ | $\pi_{m1}$ | $\pi_{m2}$ | ... | $\pi_{mn}$ |

Figure 1: Left: observed labels. Right: underlying distributions. Highlights on both tables indicate that rows and columns of the distributions are constrained by sums over observations.

objective measurements in Section 5. Finally, we present superior experimental results on real-world crowdsourcing data in Section 6.

## 2 Minimax Entropy Principle

We propose a model illustrated in Figure 1. Each row corresponds to a crowdsourced worker indexed by $i$ (from 1 to $m$). Each column corresponds to an item to be labeled, indexed by $j$ (from 1 to $n$). Each item has an unobserved label represented as a vector $y_{jl}$, which is 1 when item $j$ is in class $l$ (from 1 to $c$), and 0 otherwise. More generally, we can treat $y_{jl}$ as the probability that item $j$ is in class $l$. We observe a matrix of labels $z_{ij}$ by workers. The label matrix can also be represented as a tensor $z_{ijk}$, which is 1 when worker $i$ labels item $j$ as class $k$, and 0 otherwise. We assume that $z_{ij}$ are drawn from $\pi_{ij}$, which is the distribution for worker $i$ to generate a label for item $j$. Again, $\pi_{ij}$ can also be represented as a tensor $\pi_{ijk}$, which is the probability that worker $i$ labels item $j$ as class $k$. Our method will estimate $y_{jl}$ from the observed $z_{ij}$.

We specify the form of $\pi_{ij}$ through the maximum entropy principle, where the constraints on the maximum entropy combine the best ideas from previous work. Majority voting suggests that we should be constraining the $\pi_{ij}$ per column, with the empirical observation of the number of votes per class per item $\sum_i z_{ijk}$ should match $\sum_i \pi_{ijk}$. Dawid and Skene's method suggests that we should be constraining the $\pi_{ij}$ per row, with the empirical confusion matrix per worker $\sum_j y_{jl} z_{ijk}$ should match $\sum_j y_{jl} \pi_{ijk}$. We thus have the following maximum entropy model for $\pi_{ij}$ given $y_{jl}$:

$$\max_{\pi} \quad -\sum_{i=1}^{m}\sum_{j=1}^{n}\sum_{k=1}^{c} \pi_{ijk} \ln \pi_{ijk}$$

$$\text{s.t.} \quad \sum_{i=1}^{m}\pi_{ijk} = \sum_{i=1}^{m} z_{ijk},\ \forall j,k,\ \sum_{j=1}^{n} y_{jl}\pi_{ijk} = \sum_{j=1}^{n} y_{jl}z_{ijk},\ \forall i,k,l, \tag{1a}$$

$$\sum_{k=1}^{c}\pi_{ijk} = 1,\ \forall i,j,\ \pi_{ijk} \geq 0,\ \forall i,j,k. \tag{1b}$$

We propose that, to infer $y_{jl}$, we should choose $y_{jl}$ to *minimize* the entropy in Equation (1). Intuitively, making $\pi_{ij}$ "peaky" means that $z_{ij}$ is the least random given $y_{jl}$. We make this intuition rigorous in Section 2.1. Thus, the inference for $y_{jl}$ can be expressed by a minimax entropy program:

$$\min_{y}\max_{\pi} \quad -\sum_{i=1}^{m}\sum_{j=1}^{n}\sum_{k=1}^{c} \pi_{ijk} \ln \pi_{ijk}$$

$$\text{s.t.} \quad \sum_{i=1}^{m}\pi_{ijk} = \sum_{i=1}^{m} z_{ijk},\ \forall j,k,\ \sum_{j=1}^{n} y_{jl}\pi_{ijk} = \sum_{j=1}^{n} y_{jl}z_{ijk},\ \forall i,k,l, \tag{2a}$$

$$\sum_{k=1}^{c}\pi_{ijk} = 1,\ \forall i,j,\ \pi_{ijk} \geq 0,\ \forall i,j,k,\ \sum_{l=1}^{c} y_{jl} = 1,\ \forall j,\ y_{jl} \geq 0,\ \forall j,l. \tag{2b}$$

## 2.1 Justification for Minimum Entropy

Now we justify the principle of choosing $y_{jl}$ by minimizing entropy. Think of $y_{jl}$ as a set of parameters to the worker-item label models $\pi_{ij}$. The goal in choosing the $y_{jl}$ is to select $\pi_{ij}$ that are as close as possible to the true distributions $\pi_{ij}^*$.

To find a principle to choose the $y_{jl}$, assume that we have access to the row and column measurements on the true distributions $\pi_{ij}^*$. That is, assume that we know the true values of the column measurements $\phi_{jk} = \sum_i \pi_{ijk}^*$ and row measurements $\varphi_{ikl} = \sum_j y_{jl}\pi_{ijk}^*$, for a chosen set of $y_{jl}$ values. Knowing these true row and column measurements, we can apply the maximum entropy principle to generate distributions $\pi_{ij}$:

$$
\begin{aligned}
\max_{\pi} \quad & -\sum_{i=1}^{m}\sum_{j=1}^{n}\sum_{k=1}^{c}\pi_{ijk}\ln\pi_{ijk} \\
\text{s.t.} \quad & \sum_{i=1}^{m}\pi_{ijk} = \phi_{jk},\ \forall j,k,\ \sum_{j=1}^{n}y_{jl}\pi_{ijk} = \varphi_{ikl},\ \forall i,k,l.
\end{aligned}
\tag{3}
$$

Let $D_{\mathrm{KL}}(\cdot \parallel \cdot)$ denote the KL divergence between two distributions. We can choose $y_{jl}$ to minimize a loss of $\pi_{ij}$ with respect to $\pi_{ij}^*$ given by

$$
\ell(\pi^*,\pi) = \sum_{i=1}^{m}\sum_{j=1}^{n}D_{\mathrm{KL}}(\pi_{ij}^* \parallel \pi_{ij}).
\tag{4}
$$

The minimum loss can be attained by choosing $y_{jl}$ to minimize the entropy of the maximum distributions $\pi_{ij}$. This can be shown by writing the Lagrangian of program (3):

$$
\begin{aligned}
L = & -\sum_{i=1}^{m}\sum_{j=1}^{n}\sum_{k=1}^{c}\pi_{ijk}\ln\pi_{ijk} + \sum_{i=1}^{m}\sum_{j=1}^{n}\lambda_{ij}\left(\sum_{k=1}^{c}\pi_{ijk} - 1\right) \\
& + \sum_{j=1}^{n}\sum_{k=1}^{c}\tau_{jk}\sum_{i=1}^{m}(\pi_{ijk} - \pi_{ijk}^*) + \sum_{i=1}^{m}\sum_{k=1}^{c}\sum_{l=1}^{c}\sigma_{ikl}\sum_{j=1}^{n}y_{jl}(\pi_{ijk} - \pi_{ijk}^*),
\end{aligned}
$$

where the newly introduced variables $\tau_{jk}$ and $\sigma_{ikl}$ are the Lagrange multipliers. For a solution to be optimal, the Karush-Kuhn-Tucker (KKT) conditions must be satisfied [3]. Thus,

$$
\frac{\partial L}{\partial \pi_{ijk}} = -\ln\pi_{ijk} - 1 + \lambda_{ij} + \sum_{l=1}^{c}y_{jl}(\tau_{jk} + \sigma_{ikl}) = 0,\ \forall i,j,k,
$$

which can be rearranged as

$$
\pi_{ijk} = \exp\left[\sum_{l=1}^{c}y_{jl}(\tau_{jk} + \sigma_{ikl}) + \lambda_{ij} - 1\right],\ \forall i,j,k.
\tag{5}
$$

For being a probability measure, the variables $\pi_{ijk}$ have to satisfy

$$
\sum_{k=1}^{c}\pi_{ijk} = \sum_{k=1}^{c}\exp\left[\sum_{l=1}^{c}y_{jl}(\tau_{jk} + \sigma_{ikl}) + \lambda_{ij} - 1\right] = 1, \forall i,j.
\tag{6}
$$

Eliminating $\lambda_{ij}$ by jointly considering Equations (5) and (6), we obtain a labeling model in the exponential family:

$$
\pi_{ijk} = \frac{\exp\sum_{l=1}^{c}y_{jl}(\tau_{jk} + \sigma_{ikl})}{\sum_{s=1}^{c}\exp\sum_{l=1}^{c}y_{jl}(\tau_{js} + \sigma_{isl})},\ \forall i,j,k.
\tag{7}
$$

Plugging Equation (7) into (4) and performing some algebraic manipulations, we prove

**Theorem 2.1** *Let $\pi_{ij}$ be the maximum entropy distributions in (3). Then,*

$$
\ell(\pi^*,\pi) = \sum_{i=1}^{m}\sum_{j=1}^{n}\sum_{k=1}^{c}(\pi_{ijk}^*\ln\pi_{ijk}^* - \pi_{ijk}\ln\pi_{ijk}).
$$

The second term is the only term that depends on $y_{jl}$. Therefore, we should choose $y_{jl}$ to minimize the entropy of the maximum entropy distributions.

The labeling model expressed by Equation (7) has a natural interpretation. For each worker $i$, the multiplier set $\{\sigma_{ikl}\}$ is a measure of her expertise, while for each item $j$, the multiplier set $\{\tau_{jk}\}$ is a measure of its confusability. A worker correctly labels an item either because she has good expertise or because the item is not that confusing. When the item or worker parameters are shifted by an arbitrary constant, the probability given by Equation (7) does not change. The redundancy of the constraints in (2a) causes the redundancy of the parameters.

## 3 Constraint Relaxation

In real crowdsourcing applications, each item is usually labeled only a few times. Moreover, a worker usually only labels a small subset of items rather than all of them. In such cases, it is unreasonable to expect that the constraints in (2a) hold for the true underlying distributions $\pi_{ij}$. As in the literature of regularized maximum entropy [14, 1, 9], we relax the optimization problem to prevent overfitting:

$$\min_{y} \max_{\pi,\xi,\zeta} \quad -\sum_{i=1}^{m}\sum_{j=1}^{n}\sum_{k=1}^{c} \pi_{ijk} \ln \pi_{ijk} - \sum_{j=1}^{n}\sum_{k=1}^{c} \frac{\xi_{jk}^2}{2\alpha_j} - \sum_{i=1}^{m}\sum_{k=1}^{c}\sum_{l=1}^{c} \frac{\zeta_{ikl}^2}{2\beta_i}$$

$$\text{s.t.} \quad \sum_{i=1}^{m}(\pi_{ijk} - z_{ijk}) = \xi_{jk}, \ \forall j,k, \ \sum_{j=1}^{n} y_{jl}(\pi_{ijk} - z_{ijk}) = \zeta_{ikl}, \ \forall i,k,l, \tag{8a}$$

$$\sum_{k=1}^{c} \pi_{ijk} = 1, \ \forall i,j, \ \pi_{ijk} \ge 0, \ \forall i,j,k, \ \sum_{l=1}^{c} y_{jl} = 1, \ \forall j, \ y_{jl} \ge 0, \ \forall j,l, \tag{8b}$$

where $\alpha_j$ and $\beta_i$ are regularization parameters. It is obvious that program (8) is reduced to program (2) when the slack variables $\xi_{jk}$ and $\zeta_{ikl}$ are set to zero. The two $\ell_2$-norm based regularization terms in the objective function force the slack variables to be not far away from zero. Other vector or matrix norms, such as the $\ell_1$-norm and the trace norm, can be applied as well [14, 1, 9]. We choose the $\ell_2$-norm only for the sake of simplicity in computation.

The justification for minimum entropy in Section 2.1 can be extended to the regularized minimax entropy formulation (8) with minor modifications. Instead of knowing the exact marginals, we need to choose $\pi_{ij}$ based on noisy marginals:

$$\phi_{jk} = \sum_{i=1}^{m} \pi_{ijk}^* + \xi_{jk}^*, \forall j,k, \ \varphi_{ikl} = \sum_{j=1}^{n} y_{jl}\pi_{ijk}^* + \zeta_{ikl}^*, \forall i,k,l.$$

We thus maximize the regularized entropy subject to the relaxed constraints:

$$\sum_{i=1}^{m} \pi_{ijk} + \xi_{jk} = \phi_{jk}, \ \forall j,k, \ \sum_{j=1}^{n} y_{jl}\pi_{ijk} + \zeta_{ikl} = \varphi_{ikl}, \ \forall i,k,l. \tag{9}$$

**Lemma 3.1** *To be the regularized maximum entropy distributions subject to (9), $\pi_{ij}$ must be represented as in Equation (7). Moreover, we should have $\xi_{jk} = \alpha_j\tau_{jk}$, $\zeta_{ikl} = \beta_i\sigma_{ikl}$.*

**Proof** The first part of the result can be verified as before. By using the labeling model in Equation (7), the Lagrangian of the regularized maximum entropy program can be written as

$$L = -\sum_{i=1}^{m}\sum_{j=1}^{n} \ln \left( \sum_{s=1}^{c} \exp \sum_{l=1}^{c} y_{jl} \left( \tau_{js} + \sigma_{isl} \right) \right) - \sum_{j=1}^{n}\sum_{k=1}^{c} \frac{\xi_{jk}^2}{2\alpha_j} - \sum_{i=1}^{m}\sum_{k=1}^{c}\sum_{l=1}^{c} \frac{\zeta_{ikl}^2}{2\beta_i}$$

$$+ \sum_{j=1}^{n}\sum_{k=1}^{c} \tau_{jk} \left[ -\sum_{i=1}^{m} \pi_{ijk}^* + (\xi_{jk} - \xi_{jk}^*) \right] + \sum_{i=1}^{m}\sum_{k=1}^{c}\sum_{l=1}^{c} \sigma_{ikl} \left[ -\sum_{j=1}^{n} y_{jl}\pi_{ijk}^* + (\zeta_{ikl} - \zeta_{ikl}^*) \right].$$

For fixed $\tau_{jk}$ and $\sigma_{ikl}$, maximizing the Lagrange dual over $\xi_{jk}$ and $\zeta_{ikl}$ provides the proof.

By Lemma 3.1 and some algebraic manipulations, we obtain

**Theorem 3.2** *Let $\pi_{ij}$ be the regularized maximum entropy distributions subject to (9). Then,*

$$
\begin{aligned}
\ell(\pi^*, \pi) = & \sum_{i=1}^{m}\sum_{j=1}^{n}\sum_{k=1}^{c} \pi_{ijk}^* \ln \pi_{ijk}^* - \sum_{i=1}^{m}\sum_{j=1}^{n}\sum_{k=1}^{c} \pi_{ijk} \ln \pi_{ijk} - \sum_{j=1}^{n}\sum_{k=1}^{c} \frac{\xi_{jk}^2}{\alpha_j} \\
& - \sum_{i=1}^{m}\sum_{k=1}^{c}\sum_{l=1}^{c} \frac{\zeta_{ikl}^2}{\beta_i} + \sum_{j=1}^{n}\sum_{k=1}^{c} \frac{\xi_{jk}^* \xi_{jk}}{\alpha_j} + \sum_{i=1}^{m}\sum_{k=1}^{c}\sum_{l=1}^{c} \frac{\zeta_{ikl}^* \zeta_{ikl}}{\beta_i}.
\end{aligned}
\tag{10}
$$

We cannot minimize the loss by minimizing the right side of Equation (10) since the random noise is unknown. However, we can consider minimizing an upper bound instead. Note that

$$
\xi_{jk}^* \xi_{jk} \le (\xi_{jk}^{*2} + \xi_{jk}^2)/2, \ \forall j, k, \quad \zeta_{ikl}^* \zeta_{ikl} \le (\zeta_{ikl}^{*2} + \zeta_{ikl}^2)/2, \ \forall i, k, l.
\tag{11}
$$

Denote by $\Omega(\pi, \xi, \zeta)$ the objective function of the regularized minimax entropy program (8). Substituting the inequalities in (11) into Equation (10), we have

$$
\ell(\pi^*, \pi) \le \Omega(\pi, \xi, \zeta) - \Omega(\pi^*, \xi^*, \zeta^*).
\tag{12}
$$

So minimizing the regularized maximum entropy leads to minimizing an upper bound of the loss.

## 4 Optimization Algorithm

A typical approach to constrained optimization is to covert the primal problem to its dual form. By Lemma 3.1, the Lagrangian of program (8) can be written as

$$
L = -\sum_{j=1}^{n} \ln \left[ \prod_{i=1}^{m} \frac{\exp \sum_{k=1}^{c} z_{ijk} \sum_{l=1}^{c} y_{jl}(\tau_{jk} + \sigma_{ikl})}{\sum_{s=1}^{c} \exp \sum_{l=1}^{c} y_{jl}(\tau_{js} + \sigma_{isl})} \right] + \sum_{j=1}^{n}\sum_{k=1}^{c} \frac{\alpha_j \tau_{jk}^2}{2} + \sum_{i=1}^{m}\sum_{k=1}^{c}\sum_{l=1}^{c} \frac{\beta_i \sigma_{ikl}^2}{2}.
$$

The dual problem minimizes $L$ subject to the constraints $\Delta = \{y_{jl} | \sum_{l=1}^{c} y_{jl} = 1, \ \forall j, \ y_{jl} \ge 0, \ \forall j, l\}$. It can be solved by coordinate descent with the variables being split into two groups: $\{y_{jl}\}$ and $\{\tau_{jk}, \sigma_{ikl}\}$. It is easy to check that, when the variables in one group are fixed, the optimization problem on the variables in the other group is convex. When the $y_{jl}$ are restricted to be $\{0, 1\}$, that is, deterministic labels, the coordinate descent procedure can be simplified. Let

$$
p_{jl} = \prod_{i=1}^{m} \frac{\exp \sum_{k=1}^{c} z_{ijk}(\tau_{jk} + \sigma_{ikl})}{\sum_{s=1}^{c} \exp (\tau_{js} + \sigma_{isl})}.
$$

For any set of real-valued numbers $\{\mu_{jl} | \sum_{l=1}^{c} \mu_{jl} = 1, \ \forall j, \ \mu_{jl} > 0, \ \forall j, l\}$, we have the inequality

$$
\begin{aligned}
& \sum_{j=1}^{n} \ln \left[ \prod_{i=1}^{m} \frac{\exp \sum_{k=1}^{c} z_{ijk} \sum_{l=1}^{c} y_{jl}(\tau_{jk} + \sigma_{ikl})}{\sum_{s=1}^{c} \exp \sum_{l=1}^{c} y_{jl}(\tau_{js} + \sigma_{isl})} \right] = \sum_{j=1}^{n} \ln \sum_{l=1}^{c} y_{jl} p_{jl} \quad \text{(deterministic labels)} \\
& = \sum_{j=1}^{n} \ln \sum_{l=1}^{c} \mu_{jl} \left( \frac{y_{jl} p_{jl}}{\mu_{jl}} \right) \ge \sum_{j=1}^{n}\sum_{l=1}^{c} \mu_{jl} \ln \left( \frac{y_{jl} p_{jl}}{\mu_{jl}} \right) \quad \text{(Jensen's inequality)} \\
& = \sum_{j=1}^{n}\sum_{l=1}^{c} \mu_{jl} \ln(y_{jl} p_{jl}) - \sum_{j=1}^{n}\sum_{l=1}^{c} \mu_{jl} \ln \mu_{jl}.
\end{aligned}
$$

Plugging the last line into the Lagrangian $L$, we obtain an upper bound of $L$, called $F$. It can be shown that we must have $y_{jl} = \mu_{jl}$ at any stationary point of $F$. Our optimization algorithm is a coordinate descent minimization of this $F$ [15, 7]. We initialize $y_{jl}$ with majority vote in Equation (13). In each iteration step, we first optimize over $\tau_{jk}$ and $\sigma_{ikl}$ in (14a), which can be solved by any convex optimization procedure, and next optimize over $y_{jl}$ using a simple closed form in (14b). The optimization over $y_{jl}$ is the same as applying Bayes' theorem where the result from the last iteration is considered as a prior. This algorithm can be shown to produce only deterministic labels.

---

**Algorithm 1** Minimax Entropy Learning from Crowds

---

**input:** $\{z_{ijk}\} \in \{0,1\}^{m \times n \times c}$, $\{\alpha_j\} \in \mathbb{R}_+^n$, $\{\beta_i\} \in \mathbb{R}_+^m$

**initialization:**

$$y_{jl}^0 = \frac{\sum_{i=1}^m z_{ijl}}{\sum_{i=1}^m \sum_{k=1}^c z_{ijk}}, \ \forall j, l \tag{13}$$

**for** $t = 1, 2, \ldots$

$$\{\tau_{jk}^t, \sigma_{ikl}^t\} = \arg\min_{\tau, \sigma} \sum_{i=1}^m \sum_{j=1}^n \sum_{l=1}^c y_{jl}^{t-1} \left[ \log \sum_{s=1}^c \exp(\tau_{js} + \sigma_{isl}) - \sum_{k=1}^c z_{ijk}(\tau_{jk} + \sigma_{ikl}) \right]$$

$$+ \sum_{j=1}^n \sum_{k=1}^c \frac{\alpha_j \tau_{jk}^2}{2} + \sum_{i=1}^m \sum_{k=1}^c \sum_{l=1}^c \frac{\beta_i \sigma_{ikl}^2}{2} \tag{14a}$$

$$y_{jl}^t \propto y_{jl}^{t-1} \prod_{i=1}^m \frac{\exp \sum_{k=1}^c z_{ijk}(\tau_{jk}^t + \sigma_{ikl}^t)}{\sum_{s=1}^c \exp\left(\tau_{js}^t + \sigma_{isl}^t\right)}, \ \forall j, l \tag{14b}$$

**output:** $\{y_{jl}^t\}$

---

## 5 Measurement Objectivity Principle

The measurement objectivity principle can be roughly stated as follows: (1) a comparison of labeling confusability between two items should be independent of which particular workers are included for the comparison; (2) symmetrically, a comparison of labeling expertise between two workers should be independent of which particular items are included for the comparison. The first statement is about the objectivity of item confusability. The second statement is about the objectivity of worker expertise. In what follows, we mathematically define the measurement objectivity principle. For deterministic labels, we show that the labeling model in Equation (7) can be recovered from the measurement objectivity principle.

From Equation (7), given item $j$ in class $l$, the probability that worker $i$ labels it as class $k$ is

$$\pi_{ijkl} = \frac{\exp\left(\tau_{jk} + \sigma_{ikl}\right)}{\sum_{s=1}^c \exp\left(\tau_{js} + \sigma_{isl}\right)}. \tag{15}$$

Assume that a worker $i$ has labeled two items $j$ and $j'$ both of which are from the same class $l$. With respect to the given worker $i$, for each item, we measure the confusability for class $k$ by

$$\rho_{ijk} = \frac{\pi_{ijkl}}{\pi_{ijll}}, \quad \rho_{ij'k} = \frac{\pi_{ij'kl}}{\pi_{ij'll}}. \tag{16}$$

For comparing the item confusabilities, we compute a ratio between them. To maintain the objectivity of confusability, the ratio should not depend on whichever worker is involved in the comparison. Hence, given another worker $i'$, we should have

$$\left(\frac{\pi_{ijkl}}{\pi_{ijll}}\right) \bigg/ \left(\frac{\pi_{ij'kl}}{\pi_{ij'll}}\right) = \left(\frac{\pi_{i'jkl}}{\pi_{i'jll}}\right) \bigg/ \left(\frac{\pi_{i'j'kl}}{\pi_{i'j'll}}\right). \tag{17}$$

It is straightforward to verify that the labeling model in Equation (15) indeed satisfies the objectivity requirement given by Equation (17). We can further show that a labeling model which satisfies Equation (17) has to be expressed by Equation (15). Let us rewrite Equation (17) as

$$\frac{\pi_{ijkl}}{\pi_{ijll}} = \frac{\pi_{ij'kl}}{\pi_{ij'll}} \frac{\pi_{i'jkl}}{\pi_{i'jll}} \frac{\pi_{i'j'll}}{\pi_{i'j'kl}}.$$

Without loss of generality, choose $i' = 0$ and $j' = 0$ as the fixed references such that

$$\frac{\pi_{ijkl}}{\pi_{ijll}} = \frac{\pi_{i0kl}}{\pi_{i0ll}} \frac{\pi_{0jkl}}{\pi_{0jll}} \frac{\pi_{00ll}}{\pi_{00kl}}. \tag{18}$$

Assume that the referenced worker 0 chooses a class uniformly at random for the referenced item 0. So we have $\pi_{00ll} = \pi_{00kl} = 1/c$. Equation (18) implies $\pi_{ijkl} \propto \pi_{i0kl} \pi_{0jkl}$. Reparameterizing with

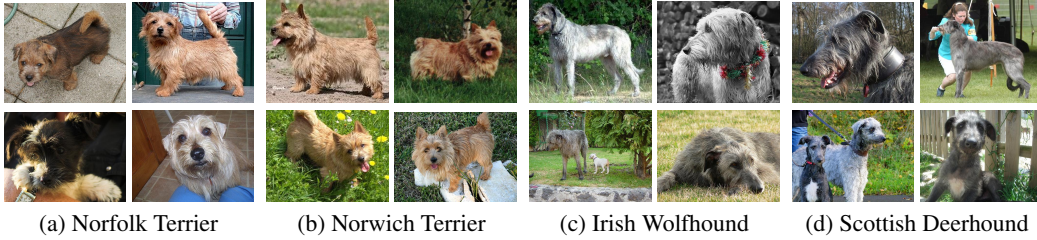

| (a) Norfolk Terrier | (b) Norwich Terrier | (c) Irish Wolfhound | (d) Scottish Deerhound |

Figure 2: Sample images of four breeds of dogs from the Stanford dogs dataset

$\pi_{i0kl} = \exp(\sigma_{ikl})$ and $\pi_{0jkl} = \exp(\tau_{jk})$ (note that $l$ is dropped since it is determined by $j$), we have $\pi_{ijkl} \propto \exp(\tau_{jk} + \sigma_{ikl})$. The labeling model in Equation (15) has been recovered.

Symmetrically, we can also start from the objectivity of worker expertise to recover the labeling model in (15). Assume that two workers $i$ and $i'$ have labeled a common item $j$ which is from class $l$. With respect to the given item $j$, for each worker, we measure the confusion from class $l$ to $k$ by

$$\rho_{ijk} = \frac{\pi_{ijkl}}{\pi_{ijll}}, \quad \rho_{i'jk} = \frac{\pi_{i'jkl}}{\pi_{i'jll}}. \tag{19}$$

For comparing the worker expertises, we compute a ratio between them. To maintain the objectivity of expertise, the ratio should not depend on whichever item is involved in the comparison. Hence, given another item $j'$ in class $l$, we should have

$$\left(\frac{\pi_{ijkl}}{\pi_{ijll}}\right) \Big/ \left(\frac{\pi_{i'jkl}}{\pi_{i'jll}}\right) = \left(\frac{\pi_{ij'kl}}{\pi_{ij'll}}\right) \Big/ \left(\frac{\pi_{i'j'kl}}{\pi_{i'j'll}}\right). \tag{20}$$

We can see that Equation (20) is actually just a rearrangement of Equation (17).

## 6 Experimental Validation

We compare our method with majority voting and Dawid & Skene's method [5] using real crowd-sourcing data. One is multiclass image labeling, and the other is web search relevance judging.

### 6.1 Image Labeling

We chose the images of 4 breeds of dogs from the Stanford dogs dataset [8]: Norfolk Terrier (172), Norwich Terrier (185), Irish Wolfhound (218), and Scottish Deerhound (232) (see Figure 2). The numbers of the images for each breed are in the parentheses. There are 807 images in total. We submitted them to MTurk, and received the labels from 109 MTurk workers. A worker labeled an image at most once, and each image was labeled 10 times. It is difficult to evaluate a worker if she only labeled few images. We thus only consider the workers who labeled at least 40 images, which yields a label set that contains 7354 labels by 52 workers. Each image has at least 4 labels and around 95% of the images have at least 8 labels. The average accuracy of the workers is 70.60%. The best worker achieved an accuracy of 88.24% while only labeled 68 images. The worker who labeled the most labeled 345 images and achieved an accuracy of 68.99%. The average worker confusion matrix between breeds is shown in Table 2. As expected, it consists of two blocks. One block contains Norfolk Terrier and Norwich Terrier, and the other block contains Irish Wolfhound and Scottish Deerhound. For our method, the regularization parameters are set as $\alpha_j = 100/(\text{number of labels for item } j)$, $\beta_i = 100/(\text{number of labels by worker } i)$. The performance of various methods on this image labeling task is summarized in Table 1. For this problem, our method is somewhat better than Dawid and Skene's method.

### 6.2 Web Search Relevance Judging

In another experiment, we asked workers to rate a set of 2665 query-URL pairs on a relevance rating scale from 1 to 5. The larger the rating, the more relevant the URL. The true labels were derived by

| Method | Dogs | Web |
|---|---|---|
| Minimax Entropy | 84.63 | 88.05 |
| Dawid & Skene | 84.14 | 83.98 |
| Majority Voting | 82.09 | 73.07 |
| Average Worker | 70.60 | 37.05 |

Table 1: Accuracy of methods (%)

| | Norfolk | Norwich | Irish | Scottish |
|---|---|---|---|---|
| Norfolk | 71.04 | 27.35 | 1.03 | 0.58 |
| Norwich | 31.99 | 66.71 | 1.13 | 0.18 |
| Irish | 1.19 | 0.55 | 69.35 | 28.91 |
| Scottish | 1.20 | 0.38 | 26.77 | 71.65 |

Table 2: Average worker confusion (%)

using consensus from 9 experts. The noisy labels were provided by 177 nonexpert workers. Each pair was judged by around 6 workers, and each worker judged a subset of the pairs. The average accuracy of workers is 37.05%. Seventeen workers have an accuracy of 0 and they judged at most 7 pairs. The worker who judged the most judged 1225 pairs and achieved an accuracy of 76.73%. For our method, the regularization parameters are set as $\alpha_j = 200/$(number of labels for item $j$), $\beta_i = 200/$(number of labels by worker $i$). The performance of various methods on this relevance judging task is summarized in Table 1. In this case, our method is substantially better.

# 7 Related Work

This paper can be regarded as a natural extension to Dawid and Skene's work [5], discussed in Section 1. Our approach can be reduced to Dawid and Skene's by setting the regularization parameters to be $\alpha_j = \infty, \beta_i = 0$. The essential difference between our work and Dawid and Skene's work is that, in addition to worker expertise, we also take item confusability into account.

In computer vision, a minimax entropy method was proposed for estimating the probability density of certain visual patterns such as textures [22]. The authors compute empirical marginal distributions through various features, then construct a density model that can reproduce all empirical marginal distributions. Among all models satisfying the constraints, the one with maximum entropy is preferred. However, one wants to select the features which are most informative: the constructed model should approximate the underlying density by minimizing a KL divergence. The authors formulate the combined density estimation and feature selection as a minimax entropy problem.

The measurement objectivity principle is inspired by the Rasch model [16], used to design and analyze psychological and educational measurements. In the Rasch model, given an examinee and a test item, the probability of a correct response is modeled as a logistic function of the difference between the examinee ability and the item difficulty. Rasch defined "specific objectivity": the comparison of any two subjects can be carried out in such a way that no other parameters are involved than those of the two subjects. The specific objectivity property of the Rasch model comes from the algebraic separation of examinee and item parameters. If the probability of a correct response is modeled with other forms, such as a logistic function of the ratio between the examinee ability and the item difficulty [21], objective measurements cannot be achieved. The most fundamental difference between the Rasch model and our work is that we must infer ground truth, rather than take them as given.

# 8 Conclusion

We have proposed a minimax entropy principle for estimating the true labels from the judgements of a crowd of nonexperts. We have also shown that the labeling model derived from the minimax entropy principle uniquely satisfies an objectivity principle for measuring worker expertise and item confusability. Experimental results on real-world crowdsourcing data demonstrate that the proposed method estimates ground truth more accurately than previously proposed methods. The presented framework can be easily extended. For example, in the web search experiment, the multilevel relevance scale is treated as multiclass. By taking the ordinal property of ratings into account, the accuracy may be further improved. The framework could be extended to real-valued labels. A detailed discussion on those topics is beyond the scope of this paper.

**Acknowledgments**

We thank Daniel Hsu, Xi Chen, Chris Burges and Chris Meek for helpful discussions, and Gabriella Kazai for generating the web search dataset.

# References

[1] Y. Altun and A. Smola. Unifying divergence minimization and statistical inference via convex duality. In *Proceedings of the 19th Annual Conference on Learning Theory*, 2006.

[2] Amazon Mechanical Turk. https://www.mturk.com/mturk.

[3] S. Boyd and L. Vandenberghe. *Convex Optimization*. Cambridge University Press, 2004.

[4] CrowdFlower. http://crowdflower.com/.

[5] A. P. Dawid and A. M. Skene. Maximum likeihood estimation of observer error-rates using the EM algorithm. *Journal of the Royal Statistical Society*, 28(1):20–28, 1979.

[6] O. Dekel and O. Shamir. Vox populi: Collecting high-quality labels from a crowd. In *Proceedings of the 22nd Annual Conference on Learning Theory*, 2009.

[7] A. P. Dempster, N. M. Laird, and D. B. Rubin. Maximum likelihood from incomplete data via the EM algorithm. *Journal of the Royal Statistical Society*, 39(1):1–38, 1977.

[8] J. Deng, W. Dong, R. Socher, L.-J. Li, K. Li, and L. Fei-Fei. ImageNet: A large-scale hierarchical image database. In *Proceedings of the IEEE Conference on Computer Vision and Pattern Recognition*, pages 248–255, 2009.

[9] M. Dudik, S. J. Phillips, and R. E. Schapire. Maximum entropy density estimation with generalized regularization and an application to species distribution modeling. *Journal of Machine Learning Research*, 8:1217–1260, 2007.

[10] S. Ertekin, H. Hirsh, and C. Rudin. Approximating the wisdom of the crowd. In *Proceedings of the Workshop on Computational Social Science and the Wisdom of Crowds*, 2011.

[11] P. G. Ipeirotis, F. Provost, and J. Wang. Quality management on Amazon Mechanical Turk. In *Proceedings of the ACM SIGKDD Workshop on Human Computation*, pages 64–67, 2010.

[12] E. Kamar, S. Hacker, and E. Horvitz. Combining human and machine intelligence in large-scale crowdsourcing. In *Proceedings of the 11th International Conference on Autonomous Agents and Multiagent Systems*, pages 467–474, 2012.

[13] D. R. Karger, S. Oh, and D. Shah. Iterative learning for reliable crowdsourcing systems. In *Advances in Neural Information Processing Systems 24*, pages 1953–1961, 2011.

[14] G. Lebanon and J. Lafferty. Boosting and maximum likelihood for exponential models. In *Advances in Neural Information Processing Systems 14*, pages 447–454, 2001.

[15] R. M. Neal and G. E. Hinton. A view of the EM algorithm that justifies incremental, sparse, and other variants. In M. I. Jordan, editor, *Learning in Graphical Models*, pages 355–368. Kluwer Academic, Dordrecht, MA, 1998.

[16] G. Rasch. On general laws and the meaning of measurement in psychology. In *Proceedings of the 4th Berkeley Symposium on Mathematical Statistics and Probability*, volume 4, pages 321–333, Berkeley, CA, 1961.

[17] V. C. Raykar, S. Yu, L. H. Zhao, G. H. Valadez, C. Florin, L. Bogoni, and L. Moy. Learning from crowds. *Journal of Machine Learning Research*, 11:1297–1322, 2010.

[18] P. Smyth, U. Fayyad, M. Burl, P. Perona, and P. Baldi. Inferring ground truth from subjective labelling of venus images. In *Advances in neural information processing systems*, pages 1085–1092, 1995.

[19] R. Snow, B. O'Connor, D. Jurafsky, and A. Y. Ng. Cheap and fast—but is it good? Evaluating non-expert annotations for natural language tasks. In *Proceedings of the Conference on Empirical Methods in Natural Language Processing*, pages 254–263, 2008.

[20] P. Welinder, S. Branson, S. Belongie, and P. Perona. The multidimensional wisdom of crowds. In *Advances in Neural Information Processing Systems 23*, pages 2424–2432, 2010.

[21] J. Whitehill, P. Ruvolo, T. Wu, J. Bergsma, and J. Movellan. Whose vote should count more: optimal integration of labels from labelers of unknown expertise. In *Advances in Neural Information Processing Systems 22*, pages 2035–2043, 2009.

[22] S. C. Zhu, Y. N. Wu, and D. B. Mumford. Minimax entropy principle and its applications to texture modeling. *Neural Computation*, 9:1627–1660, 1997.

